# Perturbing Hebbian Rules

**Peter Dayan**
CNL, The Salk Institute
PO Box 85800
San Diego CA 92186-5800, USA
dayan@helmholtz.sdsc.edu

**Geoffrey Goodhill**
COGS
University of Sussex, Falmer
Brighton BN1 9QN, UK
geoffg@cogs.susx.ac.uk

## Abstract

Recently Linsker [2] and MacKay and Miller [3, 4] have analysed Hebbian correlational rules for synaptic development in the visual system, and Miller [5, 8] has studied such rules in the case of two populations of fibres (particularly two eyes). Miller's analysis has so far assumed that each of the two populations has exactly the same correlational structure. Relaxing this constraint by considering the effects of small perturbative correlations within and between eyes permits study of the stability of the solutions. We predict circumstances in which qualitative changes are seen, including the production of binocularly rather than monocularly driven units.

## 1 INTRODUCTION

Linsker [2] studied how a Hebbian correlational rule could predict the development of certain receptive field structures seen in the visual system. MacKay and Miller [3, 4] pointed out that the form of this learning rule meant that it could be analysed in terms of the eigenvectors of the matrix of time-averaged presynaptic correlations. Miller [5, 8, 7] independently studied a similar correlational rule for the case of two eyes (or more generally two populations), explaining how cells develop in V1 that are ultimately responsive to only one eye, despite starting off as responsive to both. This process is again driven by the eigenvectors and eigenvalues of the developmental equation, and Miller [7] relates Linsker's model to the two population case.

Miller's analysis so far assumes that the correlations of activity within each population are identical. This special case simplifies the analysis enabling the projections from the two eyes to be separated out into sum and difference variables. In general, one would expect the correlations to differ slightly, and for correlations between the eyes to be not exactly zero. We analyse how such perturbations affect the eigenvectors and eigenvalues of the developmental equation, and are able to explain some of the results found empirically by Miller [6].

Further details on this analysis and on the relationship between Hebbian and non-Hebbian models of the development of ocular dominance and orientation selectivity can be found in Goodhill (1991).

## 2   THE EQUATION

MacKay and Miller [3, 4] study Linsker's [2] developmental equation in the form:
$$\dot{w} = (Q + k_2 J)w + k_1 n$$
where $w = [w_i], i \in [1, n]$ are the weights from the units in one layer $\mathcal{R}$ to a particular unit in the next layer $\mathcal{S}$, $Q$ is the covariance matrix of the activities of the units in layer $\mathcal{R}$, $J$ is the matrix $J_{ij} = 1, \forall i, j$, and $n$ is the 'DC' vector $n_i = 1, \forall i$.

The equivalent for two populations of cells is:
$$\begin{pmatrix} \dot{w_1} \\ \dot{w_2} \end{pmatrix} = \begin{pmatrix} Q_1 + k_2 J & Q_c + k_2 J \\ Q_c + k_2 J & Q_2 + k_2 J \end{pmatrix} \begin{pmatrix} w_1 \\ w_2 \end{pmatrix} + k_1 \begin{pmatrix} n \\ n \end{pmatrix}$$
where $Q_1$ gives the covariance between cells within the first population, $Q_2$ gives that between cells within the second, and $Q_c$ (assumed symmetric) gives the covariance between cells in the two populations. Define $Q_*$ as this full, two population, development matrix.

Miller studies the case in which $Q_1 = Q_2 = Q$ and $Q_c$ is generally zero or slightly negative. Then the development of $w_1 - w_2$ (which Miller calls $S^D$) and $w_1 + w_2$ ($S^S$) separate; for $Q_c = 0$, these go like:
$$\frac{\delta S^D}{\delta t} = QS^D \text{ and } \frac{\delta S^S}{\delta t} = (Q + 2k_2 J)S^S + 2k_1 n.$$
and, up to various forms of normalisation and/or weight saturation, the patterns of dominance between the two populations are determined by the initial value and the fastest growing components of $S^D$. If upper and lower weight saturation limits are reached at roughly the same time (Berns, personal communication), the conventional assumption that the fastest growing eigenvectors of $S^D$ dominate the terminal state is borne out.

The starting condition Miller adopts has $w_1 - w_2 = \epsilon' a$ and $w_1 + w_2 = b$, where $\epsilon'$ is small, and $a$ and $b$ are $\mathcal{O}(1)$. Weights are constrained to be positive, and saturate at some upper limit. Also, additive normalisation is applied throughout development, which affects the growth of the $S^S$ (but not the $S^D$) modes. As discussed by MacKay and Miller [3, 4], this is approximately accommodated in the $k_2 J$ component.

Mackay and Miller analyse the eigendecomposition of $Q + k_2 J$ for general and radially symmetric covariance matrices $Q$ and all values of $k_2$. It turns out that the eigendecomposition of $Q_*$ for the case $Q_1 = Q_2 = Q$ and $Q_c = 0$ (that studied by Miller) is given in table form by:

| E-vector | E-value | Conditions | |
|---|---|---|---|
| $(x_i, x_i)$ | $\lambda_i$ | $Qx_i = \lambda_i x_i$ | $n.x_i = 0$ |
| $(x_i, -x_i)$ | $\lambda_i$ | $Qx_i = \lambda_i x_i$ | $n.x_i = 0$ |
| $(y_i, -y_i)$ | $\mu_i$ | $Qy_i = \mu_i y_i$ | $n.y_i \neq 0$ |
| $(z_i, z_i)$ | $\nu_i$ | $(Q + 2k_2 J)z_i = \nu_i z_i$ | $n.z_i \neq 0$ |

Figure 1 shows the matrix and the two key $(y, -y)$ and $(x, -x)$ eigenvectors.

The details of the decomposition of $Q_*$ in this table are slightly obscured by degeneracy in the eigendecomposition of $Q + k_2 J$. Also, for clarity, we write $(x_i, x_i)$ for $(x_i, x_i)^T$. A consequence of the first two rows in the table is that $(\eta x_i, \theta x_i)$ is an eigenvector for any $\eta$ and $\theta$; this becomes important later.

That the development of $S^D$ and $S^S$ separates can be seen in the $(u, u)$ and $(u, -u)$ forms of the eigenvectors. In Miller's terms the onset of dominance of one of the two populations is seen in the $(u, -u)$ eigenvectors – dominance requires that $\mu_j$ for the eigenvector whose elements are all of the same sign (one such exists for Miller's $Q$) is larger than the $\mu_i$ and the $\lambda_i$ for all the other such eigenvectors. In particular, on pages 296-300 of [6], he shows various cases for which this does and one in which it does not happen. To understand how this comes about, we can treat the latter as a perturbed version of the former.

## 3   PERTURBATIONS

Consider the case in which there are small correlations between the projections and/or small differences between the correlations within each projection. For instance, one of Miller's examples indicates that small within-eye anti-correlations can prevent the onset of dominance. This can be perturbatively analysed by setting $Q_1 = Q + \epsilon E_1$, $Q_2 = Q + \epsilon E_2$ and $Q_c = \epsilon E_c$. Call the resulting matrix $Q_*^\epsilon$.

Two questions are relevant. Firstly, are the eigenvectors stable to this perturbation, *ie* are there vectors $a_1$ and $a_2$ such that $(u_1 + \epsilon a_1, u_2 + \epsilon a_2)$ is an eigenvector of $Q_*^\epsilon$ if $(u_1, u_2)$ is an eigenvector of $Q_*$ with eigenvalue $\phi$? Secondly, how do the eigenvalues change?

One way to calculate this is to consider the equation the perturbed eigenvector must satisfy:[1]

$$Q_*^\epsilon \begin{pmatrix} u_1 + \epsilon a_1 \\ u_2 + \epsilon a_2 \end{pmatrix} = (\phi + \epsilon \psi) \begin{pmatrix} u_1 + \epsilon a_1 \\ u_2 + \epsilon a_2 \end{pmatrix}$$

and look for conditions on $u_1$ and $u_2$ and the values of $a_1, a_2$ and $\psi$ by equating the $\mathcal{O}(\epsilon)$ terms. We now consider a specific example. Using the notation of the table above, $(y_i + \epsilon a_1, -y_i + \epsilon a_2)$ is an eigenvector with eigenvalue $\mu_i + \epsilon \psi_i$ if

$$(Q - \mu_i I) a_1 + k_2 J (a_1 + a_2) = -(E_1 - E_c - \psi_i I) y_i, \quad \text{and}$$
$$(Q - \mu_i I) a_2 + k_2 J (a_1 + a_2) = -(E_c - E_2 + \psi_i I) y_i.$$

Subtracting these two implies that

$$(Q - \mu_i I)(a_1 - a_2) = -(E_1 - 2E_c + E_2 - 2\psi_i I) y_i.$$

However, $y_i^T (Q - \mu_i I) = 0$, since $Q$ is symmetric and $y_i$ is an eigenvector with eigenvalue $\mu_i$, so multiplying on the left by $y_i^T$, we require that

$$2\psi_i y_i^T y_i = y_i^T (E_1 - 2E_c + E_2) y_i$$

which sets the value of $\psi_i$. Therefore $(y_i, -y_i)$ *is* stable in the required manner.

Similarly $(z_i, z_i)$ is stable too, with an equivalent perturbation to its eigenvalue. However the pair $(x_i, x_i)$ and $(x_i, -x_i)$ are not stable – the degeneracy from their having the same eigenvalue is broken, and two specific eigenvectors, $(\alpha_i x_i, \beta_i x_i)$ and $(-\beta_i x_i, \alpha_i x_i)$ are stable, for particular values $\alpha_i$ and $\beta_i$. This means that to first order, $S^D$ and $S^S$ no longer separate, and the full, two-population, matrix must be solved.

To model Miller's results, call $Q_*^{\epsilon, m}$ the special case of $Q_*^{\epsilon}$ for which $E_1 = E_2 = E$ and $E_c = 0$. Also, assume that the $x_i$, $y_i$ and $z_i$ are normalised, let $e_1(u) = u^T E_1 u$, *etc*, and define $\gamma(u) = (e_1(u) - e_2(u))/2e_c(u)$, for $e_c(u) \neq 0$, and $\gamma_i = \gamma(x_i)$. Then we have

$$\beta_i/\alpha_i = -\gamma_i \pm \sqrt{1 + \gamma_i^2} \tag{1}$$

and the eigenvalues are:

| E-vector | $Q_*$ | $Q_*^{\epsilon, m}$ | $Q_*^{\epsilon}$ |
|---|---|---|---|
| | | | **Eigenvalue for case:** |
| $(\alpha_i x_i, \beta_i x_i)$ | $\lambda_i$ | $\lambda_i + \epsilon e_1(x_i)$ | $\lambda_i + \epsilon[e_1(x_i) + e_2(x_i) + \Xi_i]/2$ |
| $(-\beta_i x_i, \alpha_i x_i)$ | $\lambda_i$ | $\lambda_i + \epsilon e_1(x_i)$ | $\lambda_i - \epsilon[e_1(x_i) + e_2(x_i) + \Xi_i]/2$ |
| $(y_i, -y_i)$ | $\mu_i$ | $\mu_i + \epsilon e_1(y_i)$ | $\mu_i + \epsilon[e_1(y_i) + e_2(y_i) - 2e_c(y_i)]/2$ |
| $(z_i, z_i)$ | $\nu_i$ | $\nu_i + \epsilon e_1(z_i)$ | $\nu_i + \epsilon[e_1(z_i) + e_2(z_i) + 2e_c(z_i)]/2$ |

where $\Xi_i = \sqrt{[e_1(x_i) - e_2(x_i)]^2 + 4e_c(x_i)^2}$. For the case Miller treats, since $E_1 = E_2$, the degeneracy in the original solution is preserved, *ie* the perturbed versions of $(x_i, x_i)$ and $(x_i, -x_i)$ have the same eigenvalues. Therefore the $S^D$ and $S^S$ modes still separate.

This perturbed eigendecomposition suffices to show how small additional correlations affect the solutions. We will give three examples. The case mentioned above on page 299 of [6], shows how small same-eye anti-correlations within the radius of the arbor function cause a particular $(y_i, -y_i)$ eigenvector (i.e. one for which all the components of $y_i$ have the same sign) to change from growing faster than a $(x_i, -x_i)$ (for which some components of $x_i$ are positive and some negative to ensure that $n.x_i = 0$) to growing slower than it, converting a monocular solution to a binocular one.

In our terms, this is the $Q_*^{\epsilon, m}$ case, with $E_1$ a negative matrix. Given the conditions on signs of their components, $e_1(y_i)$ is more negative than $e_1(x_i)$, and so the eigenvalue for the perturbed $(y_i, -y_i)$ would be expected to decrease more than that for the perturbed $(x_i, -x_i)$. This is exactly what is found. Different binocular eigensolutions are affected by different amounts, and it is typically a delicate issue as to which will ultimately prevail. Figure 2 shows a sample perturbed matrix for which dominance will not develop. If the change in the correlations is large ($\mathcal{O}(1)$), then the eigenfunctions can change shape (eg 1s becomes 2s in the notation of [4]). We do not address this here, since we are considering only changes of $\mathcal{O}(\epsilon)$.

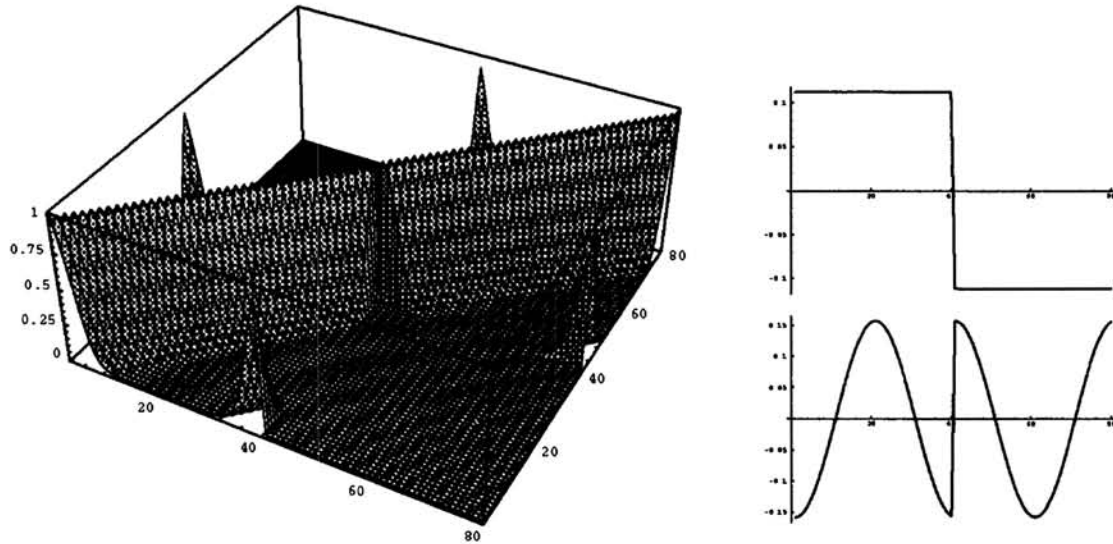

Figure 1: Unperturbed two-eye correlation matrix and $(y, -y)$, $(x, -x)$ eigenvectors. Eigenvalues are 7.1 and 6.4 respectively.

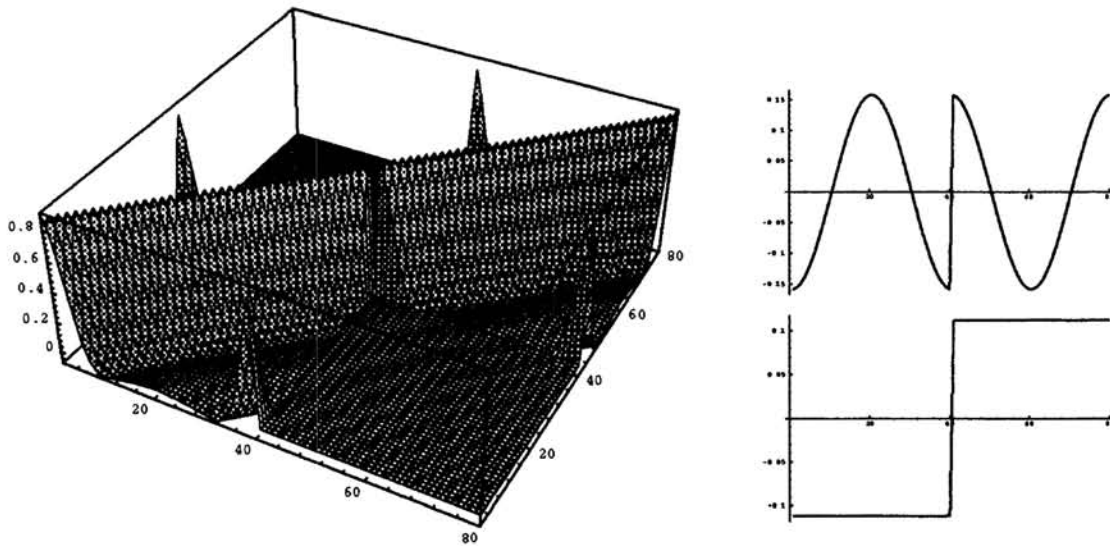

Figure 2: Same-eye anti-correlation matrix and eigenvectors. $(y, -y)$, $(x, -x)$ eigenvalues are 4.8 and 5.4 respectively, and so the order has swapped.

Positive opposite-eye correlations can have exactly the same effect. This time $e_c(y_i)$ is greater than $e_c(x_i)$, and so, again, the eigenvalue for the perturbed $(y_i, -y_i)$ would be expected to decrease more than that for the perturbed $(x_i, -x_i)$. Figure 3 shows an example which is infelicitous for dominance.

The third case is for general perturbations in $Q_*^\epsilon$. Now the mere signs of the components of the eigenvectors are not enough to predict which will be affected more. Figure 4 gives an example for which ocular dominance will still occur. Note that the $(x_i, -x_i)$ eigenvector is no longer stable, and has been replaced by one of the form $(\alpha_i x_i, \beta_i, x_i)$.

If general perturbations of the same order of magnitude as the difference between $w_1$ and $w_2$ (ie $\epsilon' \simeq \epsilon$) are applied, the $\alpha_i$ and $\beta_i$ terms complicate Miller's $S^D$ analysis to first order. Let $w_1(0) - w_2(0) = \epsilon a$ and apply $Q_*^\epsilon$ as an iteration matrix. $w_1(n) - w_2(n)$, the difference between the projections after $n$ iterations has no $\mathcal{O}(1)$ component, but two sets of $\mathcal{O}(\epsilon)$ components; $\{2\mu_i^n (a.y_i) y_i\}$, and

$$\{ \quad \lambda_i^n[1 + \epsilon(\Upsilon_i + \Xi_i)/2\lambda_i]^n (\alpha_i x_i.w_1(0) + \beta_i x_i.w_2(0))(\alpha_i - \beta_i)x_i - $$
$$\lambda_i^n[1 + \epsilon(\Upsilon_i - \Xi_i)/2\lambda_i]^n (\alpha_i x_i.w_2(0) - \beta_i x_i.w_1(0))(\alpha_i + \beta_i)x_i \quad \}$$

where $\Upsilon_i = e_1(x_i) + e_2(x_i)$. Collecting the terms in this expression, and using equation 1, we derive

$$\left\{ \lambda_i^n \left[ (\alpha_i^2 + \beta_i^2)x_i.a + 2n\frac{\Xi_i}{\lambda_i}\gamma_i\alpha_i\beta_i x_i.b \right] x_i \right\}$$

where $b = w_1(0) + w_2(0)$. The second part of this expression depends on $n$, and is substantial because $w_1(0) + w_2(0)$ is $\mathcal{O}(1)$. Such a term does not appear in the unperturbed system, and can bias the competition between the $y_i$ and the $x_i$ eigenvectors, in particular towards the binocular solutions. Again, its precise effects will be sensitive to the unperturbed eigenvalues.

## 4  CONCLUSIONS

Perturbation analysis applied to simple Hebbian correlational learning rules reveals the following:

- Introducing small anti-correlations within each eye causes a tendency toward binocularity. This agrees with the results of Miller.
- Introducing small positive correlations between the eyes (as will inevitably occur once they experience a natural environment) has the same effect.
- The overall eigensolution is not stable to small perturbations that make the correlational structure of the two eyes unequal. This also produces interesting effects on the growth rates of the eigenvectors concerned, given the initial conditions of approximately equivalent projections from both eyes.

**Acknowledgements**

We are very grateful to Ken Miller for helpful discussions, and to Christopher Longuet-Higgins for pointing us in the direction of perturbation analysis. Support

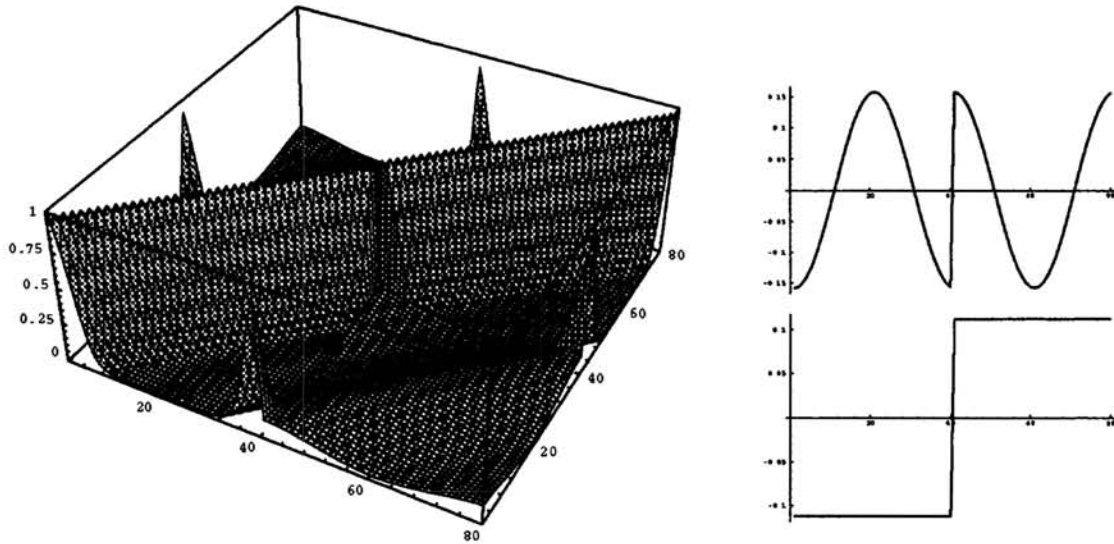

Figure 3: Opposite-eye positive correlation matrix and eigenvectors. Eigenvalues of $(y, -y)$, $(x, -x)$ are 4.8 and 5.4, so ocular dominance is again inhibited.

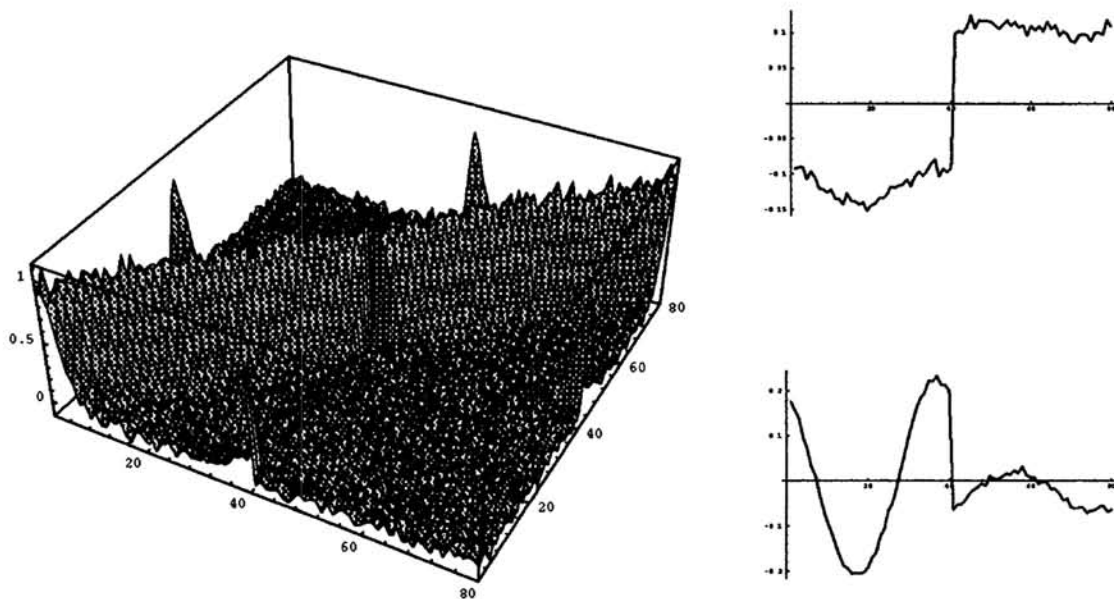

Figure 4: The effect of random perturbations to the matrix. Although the order is restored (eigenvalues are 7.1 and 6.4), note the $(\alpha x, \beta x)$ eigenvector.

was from the SERC and a Nuffield Foundation Science travel grant to GG. GG is grateful to David Willshaw and the Centre for Cognitive Science for their hospitality. GG's current address is The Centre for Cognitive Science, University of Edinburgh, 2 Buccleuch Place, Edinburgh EH8 9LW, Scotland, and correspondence should be directed to him there.

# References

[1] Goodhill, GJ (1991). *Correlations, Competition and Optimality: Modelling the Development of Topography and Ocular Dominance.* PhD Thesis, Sussex University.

[2] Linsker, R (1986). From basic network principles to neural architecture (series). *Proc. Nat. Acad. Sci., USA,* **83,** pp 7508-7512, 8390-8394, 8779-8783.

[3] MacKay, DJC & Miller, KD (1990). Analysis of Linsker's simulations of Hebbian rules. *Neural Computation,* **2,** pp 169-182.

[4] MacKay, DJC & Miller, KD (1990). Analysis of Linsker's application of Hebbian rules to linear networks. *Network,* **1,** pp 257-297.

[5] Miller, KD (1989). *Correlation-based Mechanisms in Visual Cortex: Theoretical and Empirical Studies.* PhD Thesis, Stanford University Medical School.

[6] Miller, KD (1990). Correlation-based mechanisms of neural development. In MA Gluck & DE Rumelhart, editors, *Neuroscience and Connectionist Theory.* Hillsborough, NJ: Lawrence Erlbaum.

[7] Miller, KD (1990). Derivation of linear Hebbian equations from a nonlinear Hebbian model of synaptic plasticity. *Neural Computation,* **2,** pp 321-333.

[8] Miller, KD, Keller, JB & Stryker, MP (1989). Ocular dominance column development: Analysis and simulation. *Science,* **245,** pp 605-615.
